# A Multiscale Adaptive Network Model of Motion Computation in Primates

**H. Taichi Wang**
Science Center, A18
Rockwell International
1049 Camino Dos Rios
Thousand Oaks, CA 91360

**Bimal Mathur**
Science Center, A7A
Rockwell International
1049 Camino Dos Rios
Thousand Oaks, CA 91360

**Christof Koch**
Computation & Neural Systems
Caltech, 216-76
Pasadena, CA 91125

## Abstract

We demonstrate a multiscale adaptive network model of motion computation in primate area MT. The model consists of two stages: (1) local velocities are measured across multiple spatio-temporal channels, and (2) the optical flow field is computed by a network of direction-selective neurons at multiple spatial resolutions. This model embeds the computational efficiency of Multigrid algorithms within a parallel network as well as adaptively computes the most reliable estimate of the flow field across different spatial scales. Our model neurons show the same nonclassical receptive field properties as Allman's type I MT neurons. Since local velocities are measured across multiple channels, various channels often provide conflicting measurements to the network. We have incorporated a veto scheme for conflict resolution. This mechanism provides a novel explanation for the spatial frequency dependency of the psychophysical phenomenon called Motion Capture.

## 1 MOTIVATION

We previously developed a two-stage model of motion computation in the visual system of primates (i.e. magnocellular pathway from retina to V1 and MT; Wang, Mathur & Koch, 1989). This algorithm has these deficiencies: (1) the issue of optimal spatial scale for velocity measurement, and (2) the issue optimal spatial scale for the smoothness of motion field. To address these deficiencies, we have implemented a multi-scale motion network based on multigrid algorithms.

All methods of estimating optical flow make a basic assumption about the *scale* of the velocity relative to the spatial neighborhood and to the temporal discretization step of delay. Thus, if the velocity of the pattern is much larger than the ratio of the spatial to temporal sampling step, an incorrect velocity value will be obtained (Battiti, Amaldi & Koch, 1991). Battiti *et al.* proposed a coarse-to-fine strategy for adaptively determining

the optimal discretization grid by evaluating the local estimate of the relative error in the flow field due to discretization. The optimal spatial grid is the one minimizing this error. This strategy both leads to a superior estimate of the optical flow field as well as achieving the speedups associated with multigrid methods. This is important, given the large number of iterations needed for relaxation-based algorithms and the remarkable speed with which humans can reliably estimate velocity (on the order of 10 neuronal time constants).

Our previous model was based on the standard regularization approach, which involves smoothing with weight $\lambda$. This parameter controls the smoothness of the computed motion field. The scale over which the velocity field is smooth depends on the size of the object. The larger the object is, the larger the value of $\lambda$ has to be. Since a real life vision system has to deal with objects of various sizes simultaneously, there does not exist an "optimal" smoothness parameter. Our network architecture allows us to circumvent this problem by having the same smoothing weight $\lambda$ at different resolution grids.

## 2   NETWORK ARCHITECTURE

The overall architecture of the two-stage model is shown in Figure 1. In the first stage, local velocities are measured at multiple spatial resolutions. At each spatial resolution p, the local velocities are represented by a set of direction-selective neurons, $u(i,j,k,p)$, whose preferred direction is in direction $\Theta_k$ (the Component cells; Movshon, Adelson, Gizzi & Newsome, 1985). In the second stage, the optical flow field is computed by a network of direction-selective neurons (Pattern cells) at multiple spatial resolutions, $v(i,j,k,p)$. In the following, we briefly summarize the network.

We have used a multiresolution population coding:

$$\mathbf{V} = \sum_{k}^{Nor} \sum_{p=0}^{Nres-1} \prod_{p'=p}^{1} \mathbf{I}_{p'}^{p'-1} v_k^p \, \Theta_k \,.$$

(1)

where *Nor* is the number of directions in each grid, *Nres* is the number of resolutions in the network and **I** is a 2-D linear interpolation operator (Brandt, 1982).

In our single resolution model, the input source, $s_0(i,j,k)$, to a pattern cell $v(i,j,k)$ was:

$$\frac{\partial v(i,j,k)}{\partial t} = s_0(i,j,k) = \sum_{k'} \cos(\Theta_k - \Theta_{k'}) \, \{ u(i,j,k') - (\mathbf{u} \cdot \mathbf{V}(i,j)) \} \, e(i,j,k')$$

(2)

where **u** is the the unit vector in the direction of local velocity and $e(i,j,k')$ is the local edge strength. For our multiscale network, we have used a convergent multi-channel source term, $S_0$, to a pattern cell $v(i,j,k,p)$ is:

$$s_0^p = \sum_{p' \leq p} \prod_{p''=p'}^{p} \mathbf{R}_{p''-1}^{p''} \, s_0^{p'} \,,$$

(3)

where **R** is a 2-D restriction operator. We use the full weighting operator instead of the injection operator because of the sparse nature of the input data.

The computational efficiency of the multigrid algorithms chas been embedded in our multiresolution network by a set of spatial-filtering synapses, $S_1$, written as:

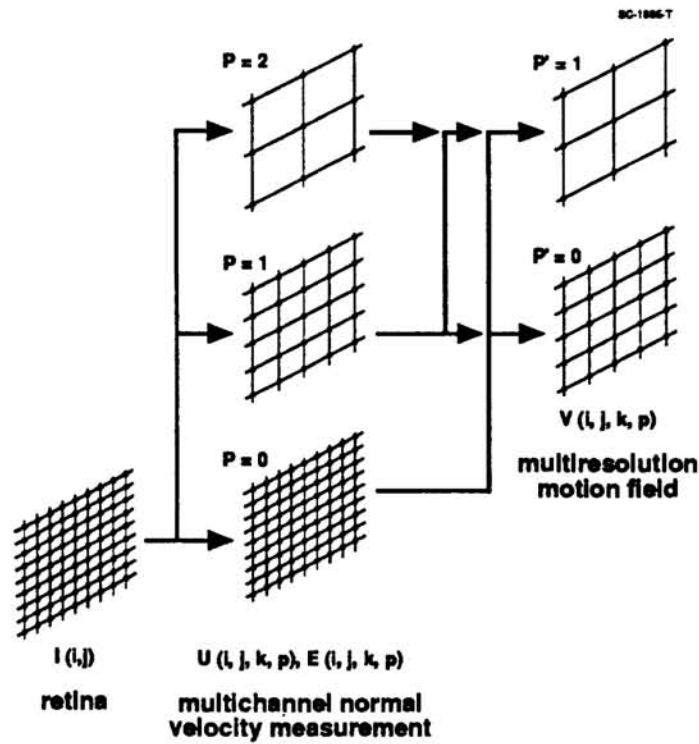

**Figure 1.** The network architecture.

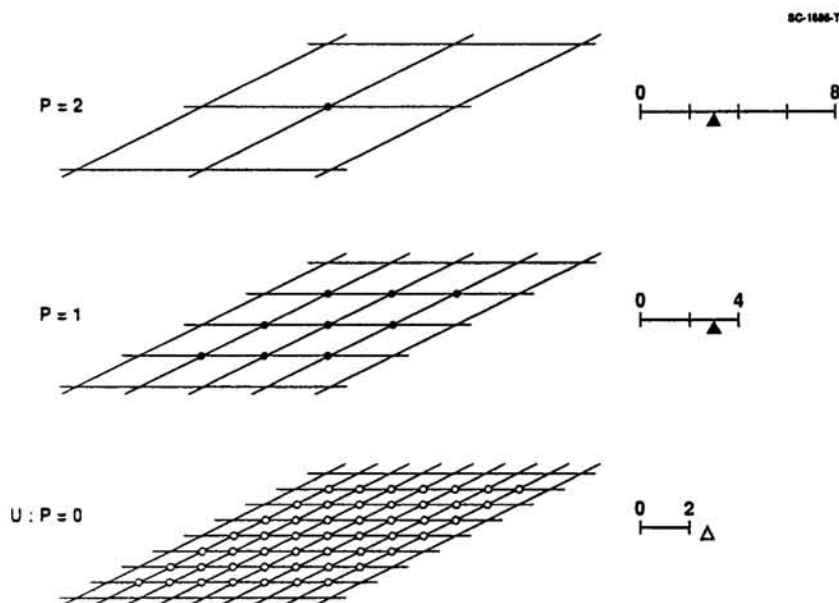

**Figure 2.** A coarse-to-fine veto scheme.

$$S_1^p = \alpha \, \mathbf{R}_{p-1}^p v^{p-1} - \beta \, \mathbf{I}_{p+1}^p \mathbf{R}_p^{p+1} \, v^p, \tag{4}$$

where $\alpha$ and $\beta$ are constants.

As discussed in the section 1, the scale over which the velocity field is smooth depends on the size of the object. Consider, for example, an object of certain size is moving with a given velocity across the field of view. The multiresolution representation and the spatial frequency filtering connections will force the velocity field to be represented mostly by a few neurons whose resolution grid matches the size of the object. Therefore, the smoothness constraint should be enforced on the individual resolution grids. If membrane potential is used, the source for the smoothness term, $S_2$, at resolution grid p, can be written as:

$$S_2^p(i,j,k) = \lambda \sum_{k'} \cos(\theta_k - \theta_{k'}) \, \{v(i-1,j,k',p) + v(i+1,j,k',p) + v(i,j-1,k',p) + v(i,j+1,k',p) - 4v(i,j,k',p)\} \tag{5}$$

where $\lambda$ is the smoothness parameter. The smoothing weight $\lambda$ in our formulation is the same for each grid and is independent of object sizes.

The network equation becomes,

$$\frac{\partial v(i,j,k,p)}{\partial t} = S_0^p + S_1^p + S_2^p. \tag{6}$$

The multiresolution network architecture has considerably more complicated synaptic connection pattern but only 33% more neurons as compared to the single resolution model, the convergence is improved by about two orders of magnitude (as measured by numbers of iterations needed).

## 3   CONFLICT RESOLUTION

The velocity estimated by our -- or any other motion algorithm -- depends on the spatial ($\Delta x$) and temporal ($\Delta t$) discretization step used. Battiti *et al.* derived the following expression for the relative error in velocity due to incorrect derivative estimation:

$$\delta = \left| \frac{\Delta u}{u} \right| \cong \frac{2\pi^2}{3\lambda^2} [(\Delta x)^2 - (u\Delta t)^2] \tag{7}$$

where u is the velocity, $\lambda$ is the spatial frequency of the moving pattern. As velocity u deviates from $\Delta x = u\Delta t$, the velocity measurement become less accurate. The scaling factor in (7) depends on the spatial filtering in the retina. Therefore, the choice of spatial discretization and spatial filtering bandwidth have to satisfy the requirements of both the *sampling theorem* and the *velocity measurement accuracy*. Even though (7) was derived based on the gradient model, we believe similar constraint applies to correlation models. We model the receptive field profiles of primate retinal ganglion cells by the Laplacian-of-Gaussian (LOG) operators. If we require that the accuracy of velocity measurement be within 10% within u = 0 to u = 2 ($\Delta x/\Delta t$), then the standard deviation, $\sigma$, of the Gaussian must be greater or equal to $2\Delta x$.

What happens if velocity measurement at various scales gives inconsistent results? Consider, for example, an object moving at a speed of 3 pixels/sec across the retina. As

shown in Figure 2, channels p=1 and p=2 will give the correct measurement, since it is in the reliable ranges of these channels, as depicted by filled circles. The finest channel, p=0, on the other hand will give an erroneous reading. This suggests a coarser-to-fine veto scheme for conflict resolution. We have incorporated this strategy in our network architecture by implementing a shunting term in Eq. (4). In this way, the erroneous input signals from the component cells at grid p=0 are shunted out (the open circles in Figure 2) by the component cells (the filled circles) at coarser grids.

## 4  MOTION CAPTURE

How does human visual system deal with the potential conflicts among various spatial channels? Is there any evidence for the use of such a coarse-to-fine conflict resolution scheme? We believe that the well-known psychophysical phenomenon of Motion Capture is the manifestation of this strategy.

When human subjects are presented a sequence of randomly moving random dots pattern, we perceive random motion. Ramachandran and Anstis (1983) found, surprisingly, that our perception of it can be greatly influenced by the movement of a superimposed low contrast, low spatial frequency grating. They found that the human subject has a tendency to perceive the random dots as moving with the spatial grating, as if the random dots adhere to the grating. For a given spatial frequency of the grating, the percentage of capture is highest when the phase shift between frames of the grating is about 90°. Even more surprisingly, the lower the spatial frequency of the grating, the higher the percentage of capture.

Other researchers (e.g. Yuille & Grzywacz, 1988) and we have attempted to explain this phenomenon based on the smoothness constraint on the velocity field. However, smoothness alone can not explain the dependencies on spatial frequency and the phase shift of the gratings. The coarser-to-fine shunting scheme provides a natural explanation of these dependencies.

We have simulated the spatial frequency and phase shift dependency. The results are shown in Figure 3. In these simulations, we plotted the relative uniformity of the motion-captured velocity fields. Uniformity of 1 signifies total capture. As can be seen clearly, for a given spatial frequency, the effect of capture increases with phase shift, and for a given phase shift, the effect of capture also increase as the spatial frequency become lower. The lower spatial frequency gratings are more effective, because the coarser the channels are, the more finer component cells can be effectively shunted out, as is clear from the receptive field relationship shown in Figure 2.

## 5  NONCLASSICAL RECEPTIVE FIELD

Traditionally, physiologists use isolated bars and slits to map out the classical receptive fields (CRF) of a neuron which is the portion of visual field that can be *directly* stimulated. Recently, there is mounting evidence that in many visual neurons stimuli presented outside the CRF strongly and selectively influence neural responses to stimuli presented within the CRF. This is termed nonclassical receptive field.

Allman, Miezin & McGuinness (1985) have found that the true receptive field of more than 90% of neurons in the middle temporal (MT) area extends well beyond their CRF. The surrounds commonly have directional and velocity-selectivity influences that are

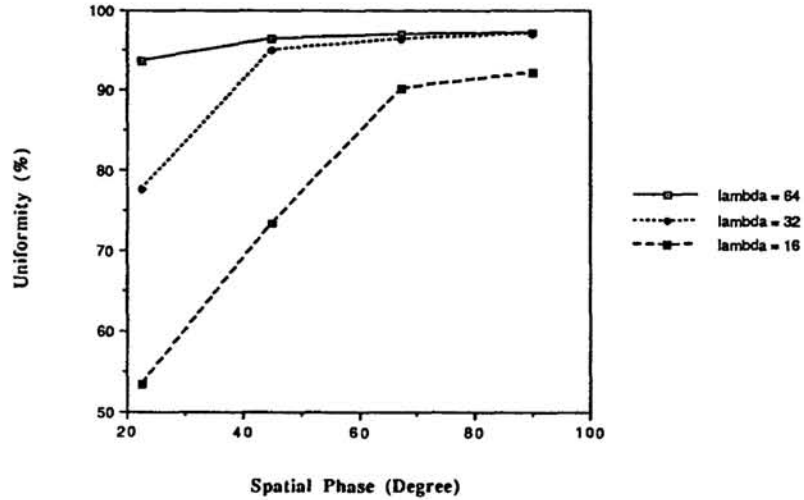

**Figure 3.** Spatial frequency dependency of Motion Capture.

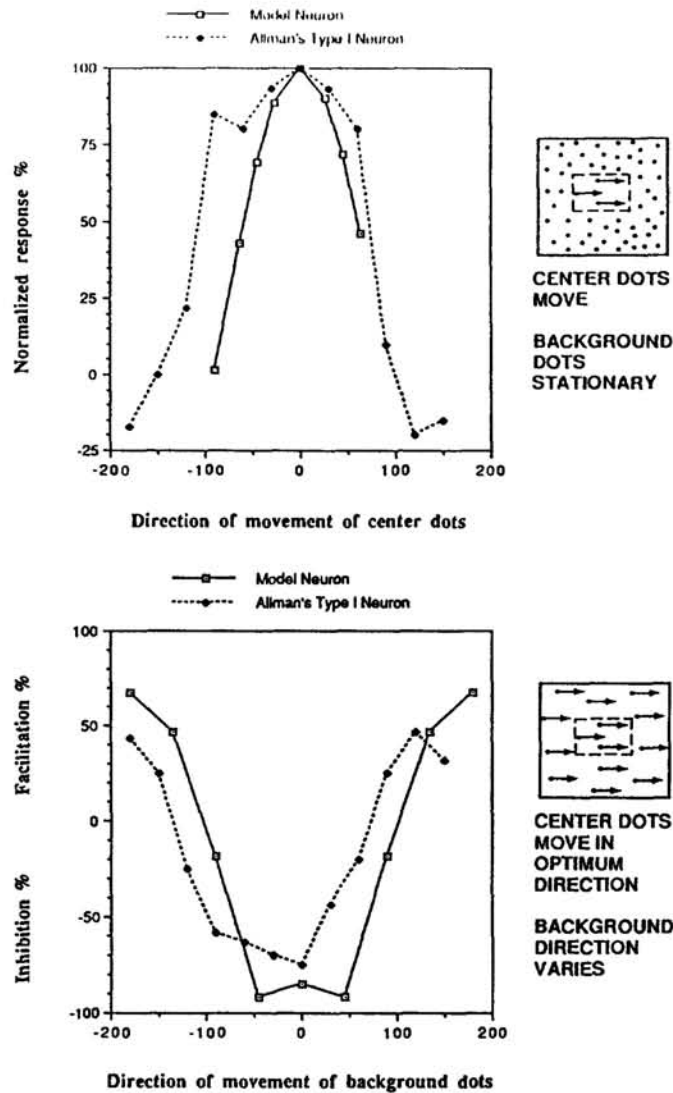

**Figure 4.** Simulation of Allman's type I non-classical receptive field properties.

antagonistic to the response from the CRF. Based on the surround selectivity, the MT neurons can be classified into three types. Our model neurons show that same type of nonclassical receptive field selectivity as Allman's type I neuron. We have performed a series of simulations similar to Allman's original experiments.

After the CRF of a model is determined, the optimal motion stimulus is presented within the CRF. The surrounds are, however, moved by the same amount but in the various directions. Clearly, the motion in the surround has profound effect of the activity of the cell we are monitoring. The effect of the surround motion on the cell as a function of the the direction of surround motion is plotted in Figure 4 (b). When the surround is moved in a similar direction as the center, the neuron activity of the cell is almost totally suppressed. On the other hand, when the surround is moved opposite to the center, the cell's activity is enhanced. Superimposed on Figure 4 are the similar plots from Allman's paper.

# 6  CONCLUSION

In conclusion, we have developed a multi-channel, multi-resolution network model of motion computation in primates. The model MT neurons show similar nonclassical surround properties as Allman's type I cells. We also proposed a novel explanation of the Motion Capture phenomenon based on a coarse-to-fine strategy for conflict resolution among the various input channels.

**Acknowledgements**

CK acknowledges ONR, NSF and the James McDonnell Foundation for supporting this research.

**References**

Allman, J., Miezin, F., and McGuinness, E. (1985) "Direction- and velocity-specific responses from beyond the classical receptive field in the middle temporal visual area (MT)", *Perception,* **14,** 105 - 126.

Battiti, R., Koch, C. and Amaldi, E. (1991) "Computing optical flow across multiple scales: an adaptive coarse-to-fine approach", to appear in *Intl. J. Computer Vision.*

Brandt, A. (1982) "Guide to multigrid development". In: *Muitlgrid Methods,* Ed. Dold, A. and Eckmann, B., Springer-Verlag.

Movshon, J.A., Adelson, E.H., Gizzi, M.S., and Newsome, W.T. (1985) "The Analysis of Moving Visual Pattern", In *Pattern Recognition Mechanisms,* ed. Chagas, C., Gattas, R., Gross, C.G., Rome: Vatican Press.

Ramachandran, V.S. and Anstis, S.M. (1983) "Displacement thresholds for coherent apparent motion in random dot-patterns", *Vision Res.* 23 (12), 1719 - 1724.

Yuille, A.L. and Grzywacz, N.M. (1988) "A computational theory for the perception of coherent visual motion", Nature, **333,** 71 - 74.

Wang, H. T., Mathur, B. P. and Koch, C. (1989) "Computing optical flow in the primate visual system", *Neural Computation,* **1**(1), 92 - 103.